# $\varepsilon$-Entropy and the Complexity of Feedforward Neural Networks

**Robert C. Williamson**
Department of Systems Engineering
Research School of Physical Sciences and Engineering
Australian National University
GPO Box 4, Canberra, 2601, Australia

## Abstract

We develop a new feedforward neural network representation of Lipschitz functions from $[0, \rho]^n$ into $[0, 1]$ based on the level sets of the function. We show that

$$\frac{n\rho L}{2\varepsilon_r} + \frac{1}{\sqrt{2}\varepsilon_r} + \left(1 + \frac{n}{\sqrt{2}}\right)\left(\frac{\rho L}{4\varepsilon_r}\right)^n$$

is an upper bound on the number of nodes needed to represent $f$ to within uniform error $\varepsilon_r$, where $L$ is the Lipschitz constant. We also show that the number of bits needed to represent the weights in the network in order to achieve this approximation is given by

$$O\left(\frac{n^2 \rho L}{\sqrt{2}\,4^n \varepsilon_r}\left(\frac{\rho L}{\varepsilon_r}\right)^n\right).$$

We compare this bound with the $\varepsilon$-entropy of the functional class under consideration.

## 1   INTRODUCTION

We are concerned with the problem of the number of nodes needed in a feedforward neural network in order to represent a function to within a specified accuracy. All results to date (e.g. [7, 10, 15]) have been in the form of existence theorems, stating that there does exist a neural network which achieves a certain accuracy of representation, but no indication is given of the number of nodes necessary in order to achieve this. The two techniques we use are the notion of $\varepsilon$-entropy (also known

Table 1: Hierarchy of theoretical problems to be solved.

ABSTRACT

1. Determination of the general approximation properties of feedforward neural networks. (Non-constructive results of the form mentioned above [15].)
2. Explicit constructive approximation theorems for feedforward neural networks, indicating the number (or bounds on the number) of nodes needed to approximate a function from a given class to within a given accuracy. (This is the subject of the present paper. We are unaware of any other work along these lines apart from [6].)
3. Learning in general. That is, results on learning that are not dependent on the particular representation chosen. The exciting new results using the Vapnik-Chervonenkis dimension [4, 9] fit into this category, as do studies on the use of Shortest Description Length principles [2].
4. Specific results on capabilities of learning in a given architecture [11].
5. Specific algorithms for learning in a specific architecture [14].

CONCRETE

as metric entropy) originally introduced by Kolmogorov [16] and a representation of a function in terms of its level sets, which was used by Arnold [1]. The place of the current paper with respect to other works in the literature can be judged from table 1.

We study the question of representing a function $f$ in the class $F_{L,C}^{(\rho_1,\ldots,\rho_n),n}$, which is the space of real valued functions defined on the $n$-dimensional closed interval $\times_{i=1}^{n}[0,\rho_i]$ with a Lipschitz constant $L$ and bounded in absolute value by $C$. If $\rho_i = \rho$ for $i = 1,\ldots,n$, we denote the space $F_{L,C}^{\rho,n}$. The error measure we use is the uniform or sup metric:

$$\varepsilon = \sup_{x \in [0,\rho]^n} |\tilde{f}(x) - f(x)|, \tag{1}$$

where $\tilde{f}$ is the approximation of $f$.

## 2    ε-ENTROPY OF FUNCTIONAL CLASSES

The ε-entropy $\mathcal{H}_\varepsilon$ gives an indication of the number of bits required to represent with accuracy $\varepsilon$ an *arbitrary* function $f$ in some functional class. It is defined as the logarithm to base 2 of the number of elements in the smallest ε-cover of the functional class. Kolmogorov [16] has proved that

$$\mathcal{H}_\varepsilon\left(F_{L,C}^{\rho,n}\right) = B(n)\left(\frac{\rho L}{\varepsilon}\right)^n \tag{2}$$

where $B(n)$ is a constant which depends only on $n$. We use this result as a yardstick for our neural network representation. A more powerful result is [18, p.86]:

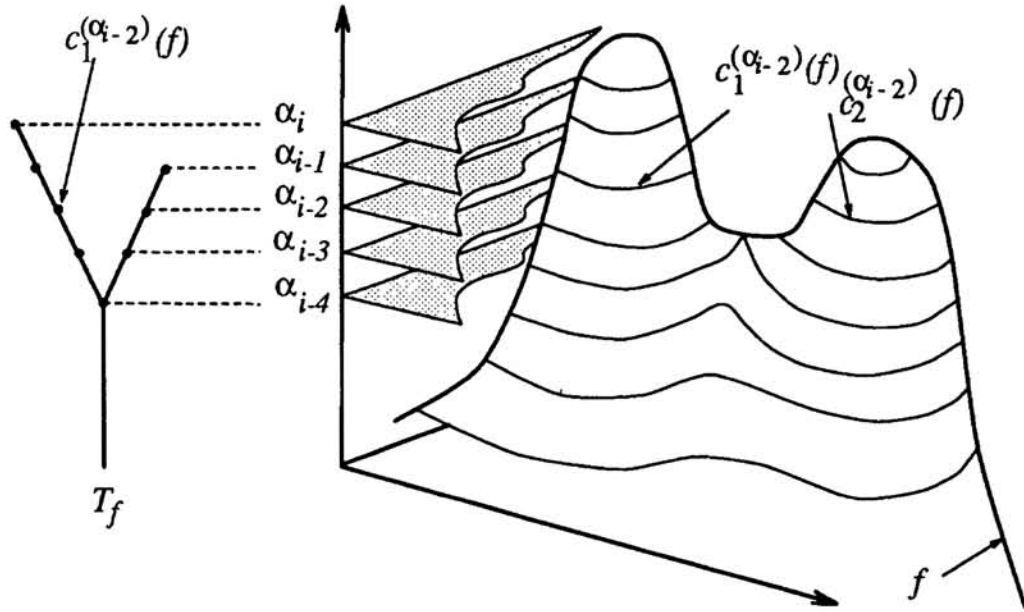

Figure 1: Illustration of some level sets of a function on $\mathbf{R}^2$.

**Theorem 1** *Let $p$ be a non-negative integer and let $\alpha \in (0,1]$. Set $s = p + \alpha$. Let $F^{\rho,n}_{s,L,C(0)}$ denote the space of real functions $f$ defined on $[0,\rho]^n$ all of whose partial derivatives of order $p$ satisfy a Lipschitz condition with constant $L$ and index $\alpha$, and are such that*

$$\left| \frac{\partial^{k_1+k_2+\cdots+k_n} f(0)}{\partial x_1^{k_1} \partial x_2^{k_2} \cdots \partial x_n^{k_n}} \right| \leq C \quad for \quad \sum_{i=1}^{n} k_i \leq p. \tag{3}$$

*Then for sufficiently small $\varepsilon$,*

$$A(s,n)\rho^n \left(\frac{L}{\varepsilon}\right)^{\frac{n}{s}} \leq \mathcal{H}_\varepsilon \left(F^{\rho,n}_{s,L,C(0)}\right) \leq B(s,n)\rho^n \left(\frac{L}{\varepsilon}\right)^{\frac{n}{s}}, \tag{4}$$

*where $A(s,n)$ and $B(s,n)$ are positive constants depending only on $s$ and $n$.*

We discuss the implication of this below.

## 3    A NEURAL NETWORK REPRESENTATION BASED ON LEVEL SETS

We develop a new neural network architecture for representing functions from $[0,\rho]^n$ onto $[0,1]$ (the restriction of the range to $[0,1]$ is just a convenience and can be easily dropped). The basic idea is to represent approximations $\tilde{f}$ of the function $f$ in terms of the level sets of $f$ (see figure 1). Then neural networks are used to approximate the *above sets* $\bar{l}_\alpha(f)$ of $f$, where $\bar{l}_\alpha(f) \triangleq \{x: f(x) \geq \alpha\} = \bigcup_{\beta \geq \alpha} l_\beta(f)$ and $l_\alpha(f)$ is the $\alpha$th level set: $l_\alpha(f) \triangleq \{x: f(x) = \alpha\}$. The approximations $\tilde{l}_\alpha(f)$ can

be implemented using three layer neural nets with threshold logic neurons. These approximations are of the form

$$\tilde{I}_{\alpha_i}(f) = \overbrace{\bigcup_{m=1}^{C_{\alpha_i}} \overbrace{\bigcup_{\lambda_m=1}^{\Lambda_m} \underbrace{\bigcap_{j=1}^{n} \left[ S(h_{U_j,\theta_j^{\lambda_m}}) \cap S(h_{-U_j,-(\theta_j^{\lambda_m}+\psi_j^{\lambda_m})}) \right]}_{n\text{-rectangle of dimensions } \psi_1^{\lambda_m} \times \cdots \times \psi_n^{\lambda_m}}}^{\text{Isothetic approximation to the } m\text{th component of } \tilde{I}_{\alpha_i}(f).}} , \tag{5}$$

where $\psi_j^{\lambda_m}$ is the "width" in the $j$th dimension of the $\lambda_m$th rectangular *part* of the $m$th *component* (disjoint connected subset) $\tilde{c}_m^{(\alpha_i)}$ of the $i$th approximate above-set $\tilde{I}_{\alpha_i}$, $C_{\alpha_i}$ is the number of components of the above-set $\tilde{I}_{\alpha_i}(f)$, $\Lambda_m$ is the number of *convex $n$-rectangles (parts)* that are required to form an $\varepsilon_l$-cover for $c_m^{(\alpha_i)}(f)$, $U_j \triangleq (u_j^{(1)},\ldots,u_j^{(n)})$, $u_j^{(m)} = \delta_{jm}$, $S(h_{w,\theta})$ is the $n$-half-space defined by the hyperplane $h_{w,\theta}$:

$$S(h_{w,\theta}) = \{x : h_{w,\theta}(x) \geq 0\}, \tag{6}$$

where $h_{w,\theta}(x) = w.x - \theta$ and $w = (w_1,\ldots,w_n)$.

The function $f$ is then approximated by

$$\tilde{f}^{N\text{-uas}}(x) \triangleq \frac{1}{2N} + \frac{1}{N} \sum_{i=1}^{N} \mathbf{1}_{\tilde{I}_{\alpha_i}(f)}(x), \tag{7}$$

where $\alpha_i = \frac{i-1}{N}$, $i = 1,\ldots,N$ and $\mathbf{1}_S$ is the indicator function of a set $S$. The approximation $\tilde{f}^{N\text{-uas}}(x)$ is then further approximated by implementing (5) using $N$ 3-layer neural nets in parallel:

$$\tilde{f}^{NN}(x) = \frac{1}{2N} + \underbrace{\sum_{i=1}^{N} s_\alpha}_{\text{last}} \underbrace{\bigvee_{m=1}^{\nu_2^{(i)}}}_{\text{third}} \underbrace{\bigwedge_{k_m=1}^{K_m^{(i)}}}_{\text{second}} \underbrace{\text{sgn}\left( \sum_{q=1}^{n} w_{k_m,q}^{(i)} x_q - \theta_{k_m}^{(i)} \right)}_{\text{first}} \quad x \in \times_{i=1}^{n}[0,\rho_i], \tag{8}$$

where $x = (x_1,\ldots,x_n)^T$, $s_\alpha = \frac{1}{N}$ and $\nu_2^{(i)}$ is the number of nodes in the second layer. The last layer combines the above-sets in the manner of (7). The general architecture of the network is shown in figure 2.

# 4  NUMBER OF BITS NEEDED TO REPRESENT THE WEIGHTS OF THE NETWORK

The two main results of this paper are bounds on the number of nodes needed in such a neural network in order to represent $f \in F_{L,C}^{\rho,n}$ with uniform error $\varepsilon_r$, and bounds on the number of bits needed to represent the weights in such an approximation.

**Theorem 2** *The number of nodes needed in a neural network of the above architecture in order to represent any $f \in F_{L,C}^{\rho,n}$ to within $\varepsilon_r$ in the sup-metric is given by*

$$\frac{n\rho L}{2\varepsilon_r} + \frac{1}{\sqrt{2}\varepsilon_r} + \left(1 + \frac{n}{\sqrt{2}}\right)\left(\frac{\rho L}{4\varepsilon_r}\right)^n . \tag{9}$$

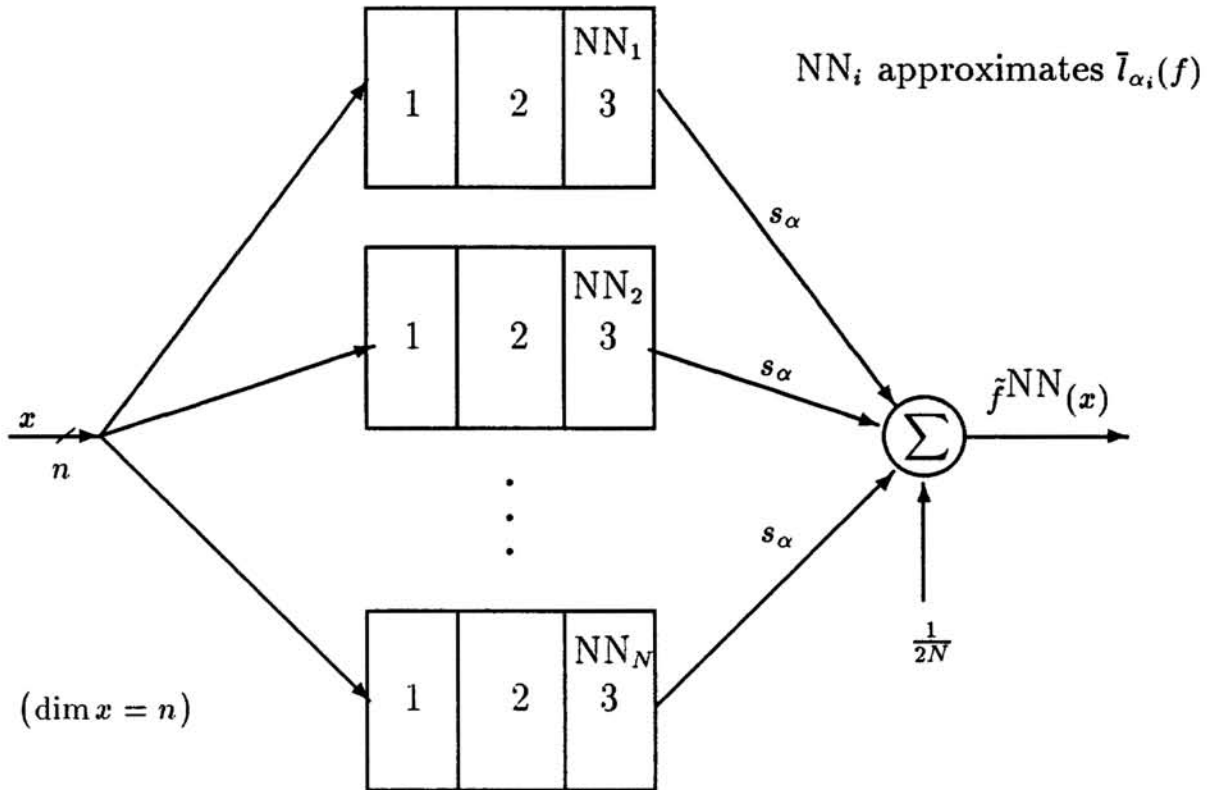

Figure 2: The Neural Network architecture we adopt.

This theorem is proved in a straight-forward manner by taking account of all the errors incurred in the approximation of a worst-case function in $F_{L,C}^{\rho,n}$.

Since comparing the number of nodes alone is inadequate for comparing the complexity of neural nets (because the nodes themselves could implement quite complex functions) we have also calculated the number of bits needed to represent all of the weights (including zero weights which denote no connection) in order to achieve an $\varepsilon_r$-approximation:[1]

**Theorem 3** *The number of bits needed to specify the weights in a neural network with the above architecture in order to represent an arbitrary function $f \in F_{L,C}^{\rho,n}$ with accuracy $\varepsilon_r$ in the sup-metric is bounded above by*

$$O\left(\frac{n^2 \rho L}{\sqrt{2}\, 4^n \varepsilon_r} \left(\frac{\rho L}{\varepsilon_r}\right)^n\right). \tag{10}$$

Equation 10 can be compared with (2) to see that the neural net representation is close to optimal. It is suboptimal by a factor of $O(\frac{\rho L}{\varepsilon})$. The $\frac{n^2}{\sqrt{2}\,4^n}$ term is considered subsumed into the $B(n)$ term in (2).

# 5   FURTHER WORK

Theorem 3 shows that the complexity of representing an arbitrary $f \in F_{L,C}^{\rho,n}$ is exponential in $n$. This is not so much a limitation of the neural network as an indication that our problem is too hard. Theorem 1 shows that if smoothness constraints are imposed, then the complexity can be considerably reduced. It is an open problem to determine whether the construction of the network presented in this paper can be extended to make good use of smoothness constraints.

Of course the most important question is whether functions can be *learned* using neural networks. Apropos of this is Stone's result on rates of convergence in nonparametric regression [17]. Although we do not have space to give details here, suffice it say that he shows that the gains suggested by theorem 1 by imposing smoothness constraints in the representation problem, are also achievable in the learning problem. A more general statement of this type of result, making explicit the connexion with ε-entropy is given by Yatracos [19]:

**Theorem 4** *Let $M$ be a $L_1$-totally bounded set of measures on a probability space. Let the metric defined on the space be the $L_1$-distance between measures. Then there exists a uniformly consistent estimator $\hat{\theta}_i$ for some parameter $\theta$ from a possibly infinite dimensional family of measures $\Theta \subset M$ whose rate of convergence in $i$ asymptotically satisfies the equation*

$$a_i = \left[ \frac{\mathcal{H}_{a_i}(\Theta)}{i} \right]^{1/2} \tag{11}$$

*where $\mathcal{H}_\varepsilon(\Theta)$ is the ε-entropy of $\Theta$.*

Similar results have been discussed by Ben-David *et al.* [3] (who have made use of Dudley's (loose) relationships between ε-entropy and Vapnik-Chervonenkis dimension [8]) and others [12, 13]. There remain many open problems in this field. One of the main difficulties however is the calculation of $\mathcal{H}_\varepsilon$ for non-trivial function classes. One of the most significant results would be a complete and tight determination of the ε-entropy for a feedforward neural network.

### Acknowledgements

This work was supported in part by a grant from ATERB. I thank Andrew Paice for many useful discussions.

## Footnotes

[1]The idea of using the number of bits as a measure of network complexity has also recently been adopted in [5].

### References

[1]   V. I. Arnold, Representation of Continuous Functions of Three Variables by the Superposition of Continuous Functions of Two Variables, *Matematicheshii Sbornik (N.S.)*, **48** (1959), pp. 3–74, Translation in *American Mathematical Society Translations Series 2*, **28** (1959) pp. 61–147.

[2]   A. R. Barron, Statistical Properties of Artificial Neural Networks, in *Proceedings of the 28th Conference on Decision and Control*, 1989, pp. 280–285.

[3]   S. Ben-David, A. Itai and E. Kushilevitz, Learning by Distances, in *Proceedings of the Third Annual Workshop on Computational Learning Theory*, M. Fulk and J. Case, eds., Morgan Kaufmann, San Mateo, 1990, pp. 232–245.

[4]   A. Blumer, A. Ehrenfeucht, D. Haussler and M. K. Warmuth, Learnability and the Vapnik-Chervonenkis Dimension, *Journal of the Association for Computing Machinery*, **36** (1989), pp. 929–965.

[5]   J. Bruck and J. W. Goodman, On the Power of Neural Networks for Solving Hard Problems, *Journal of Complexity*, **6** (1990), pp. 129–135.

[6]   S. M. Carroll and B. W. Dickinson, Construction of Neural Nets using the Radon Transform, in *Proceedings of the International Joint Conference on Neural Networks*, 1989, pp. 607–611, (Volume I).

[7]   G. Cybenko, Approximation by Superpositions of a Sigmoidal Function, *Mathematics of Control, Signals, and Systems*, **2** (1989), pp. 303–314.

[8]   R. M. Dudley, A Course on Empirical Processes, in *École d'Été de Probabilités de Saint-Flour XII-1982*, R. M. Dudley, H. Kunitay and F. Ledrappier, eds., Springer-Verlag, Berlin, 1984, pp. 1–142, Lecture Notes in Mathematics **1097**.

[9]   A. Ehrenfeucht, D. Haussler, M. Kearns and L. Valiant, A General Lower Bound on the Number of Examples Needed for Learning, *Information and Computation*, **82** (1989), pp. 247–261.

[10]  K. -I. Funahashi, On the Approximate Realization of Continuous Mappings by Neural Networks, *Neural Networks*, **2** (1989), pp. 183–192.

[11]  S. I. Gallant, A Connectionist Learning Algorithm with Provable Generalization and Scaling Bounds, *Neural Networks*, **3** (1990), pp. 191–201.

[12]  S. van de Geer, A New Approach to Least-Squares Estimation with Applications, *The Annals of Statistics*, **15** (1987), pp. 587–602.

[13]  R. Hasminskii and I. Ibragimov, On Density Estimation in the View of Kolmogorov's Ideas in Approximation Theory, *The Annals of Statistics*, **18** (1990), pp. 999–1010.

[14]  R. Hecht-Nielsen, Theory of the Backpropagation Neural Network, in *Proceedings of the International Joint Conference on Neural Networks*, 1989, pp. 593–605, Volume 1.

[15]  K. Hornik, M. Stinchcombe and H. White, Multilayer Feedforward Networks are Universal Approximators, *Neural Networks*, **2** (1989), pp. 359–366.

[16]  A. N. Kolmogorov and V. M. Tihomirov, $\varepsilon$-Entropy and $\varepsilon$-Capacity of Sets in Functional Spaces, *Uspehi Mat. (N.S.)*, **14** (1959), pp. 3–86, Translation in *American Mathematical Society Translations, Series 2*, **17** (1961) pp. 277–364.

[17]  C. J. Stone, Optimal Global Rates of Convergence for Nonparametric Regression, *The Annals of Statistics*, **10** (1982), pp. 1040–1053.

[18]  A. G. Vitushkin, *Theory of the Transmission and Processing of Information*, Pergamon Press, Oxford, 1961, Originally published as *Otsenka slozhnosti zadachi tabulirovaniya* (Estimation of the Complexity of the Tabulation Problem), Fizmatgiz, Moscow, 1959.

[19]  Y. G. Yatracos, Rates of Convergence of Minimum Distance Estimators and Kolmogorov's Entropy, *The Annals of Statistics*, **13** (1985), pp. 768–774.
